# Second order approximations for probability models

**Hilbert J. Kappen**
Department of Biophysics
Nijmegen University
Nijmegen, the Netherlands
bert@mbfys.kun.nl

**Wim Wiegerinck**
Department of Biophysics
Nijmegen University
Nijmegen, the Netherlands
wimw@mbfys.kun.nl

## Abstract

In this paper, we derive a second order mean field theory for directed graphical probability models. By using an information theoretic argument it is shown how this can be done in the absense of a partition function. This method is a direct generalisation of the well-known TAP approximation for Boltzmann Machines. In a numerical example, it is shown that the method greatly improves the first order mean field approximation. For a restricted class of graphical models, so-called single overlap graphs, the second order method has comparable complexity to the first order method. For sigmoid belief networks, the method is shown to be particularly fast and effective.

## 1 Introduction

Recently, a number of authors have proposed deterministic methods for approximate inference in large graphical models. The simplest approach gives a lower bound on the probability of a subset of variables using Jenssen's inequality (Saul et al., 1996). The method involves the minimization of the KL divergence between the target probability distribution $p$ and some 'simple' variational distribution $q$. The method can be applied to a large class of probability models, such as sigmoid belief networks, DAGs and Boltzmann Machines (BM).

For Boltzmann-Gibbs distributions, it is possible to derive the lower bound as the first term in a Taylor series expansion of the free energy around a factorized model. The free energy is given by $-\log Z$, where $Z$ is the normalization constant of the Boltzmann-Gibbs distribution: $p(x) = \frac{\exp(-E(x))}{Z}$. This Taylor series can be continued and the second order term is known as the TAP correction (Plefka, 1982; Kappen and Rodríguez, 1998). The second order term significantly improves the quality of the approximation, but is no longer a bound.

For probability distributions that are not Boltzmann-Gibbs distributions, it is not obvious how to obtain the second order approximation. However, there is an alternative way to compute the higher order corrections, based on an information theoretic argument. Recently, this argument was applied to stochastic neural networks with asymmetric connectivity (Kappen and Spanjers, 1999). Here, we apply this idea to directed graphical models.

## 2 The method

Let $x = (x_1, \ldots, x_n)$ be an $n$-dimensional vector, with $x_i$ taking on discrete values. Let $p(x)$ be a directed graphical model on $x$. We will assume that $p(x)$ can be written as a product of potentials in the following way:

$$p(x) = \prod_{k=1}^{n} p_k(x_k|\pi_k) = \exp \sum_{k=1}^{n} \phi_k(x^k). \tag{1}$$

Here, $p_k(x_k|\pi_k)$ denotes the conditional probability table of variable $x_k$ given the values of its parents $\pi_k$. $x^k = (x_k, \pi_k)$ denotes the subset of variables that appear in potential $k$ and $\phi_k(x^k) = \log p_k(x_k|\pi_k)$. Potentials can be overlapping, $x^k \cap x^l \neq \emptyset$, and $x = \cup_k x^k$.

We wish to compute the marginal probability that $x_i$ has some specific value $s_i$ in the presence of some evidence. We therefore denote $x = (e, s)$ where $e$ denote the subset of variables that constitute the evidence, and $s$ denotes the remainder of the variables. The marginal is given as

$$p(s_i|e) = \frac{p(s_i, e)}{p(e)}. \tag{2}$$

Both numerator and denominator contain sums over hidden states. These sums scale exponentially with the size of the problem, and therefore the computation of marginals is intractable.

We propose to approximate this problem by using a mean field approach. Consider a factorized distribution on the hidden variables $h$:

$$q(s) = \prod_i q_i(s_i) \tag{3}$$

We wish to find the factorized distribution $q$ that best approximates $p(s|e)$. Consider as a distance measure

$$KL = \sum_s p(s|e) \log \left( \frac{p(s|e)}{q(s)} \right). \tag{4}$$

It is easy to see that the $q$ that minimizes $KL$ satisfies:

$$q(s_i) = p(s_i|e) \tag{5}$$

We now think of the manifold of all probability distributions of the form Eq. 1, spanned by the coordinates $\phi_k(x^k), k = 1, \ldots, m$. For each $k$, $\phi_k(x^k)$ is a table of numbers, indexed by $x^k$. This manifold contains a submanifold of factorized probability distributions in which the potentials factorize: $\phi_k(x^k) = \sum_{i, i \in k} \phi_{ki}(x_i)$. When in addition, $\sum_{k, i \in k} \phi_{ki}(x_i) = \log q_i(x_i), i \in h$, $p(s|e)$ reduces to $q(s)$.

Assume now that $p(s|e)$ is somehow close to the factorized submanifold. The difference $\Delta p(s_i|e) = p(s_i|e) - q_i(s_i)$ is then small, and we can expand this small difference in terms of changes in the parameters $\Delta\phi_k(x^k) = \phi_k(x^k) - \log q(x^k), k = 1, \ldots, m$:

$$\begin{aligned} \Delta \log p(s_i|e) &= \sum_{k=1}^{n} \sum_{\bar{x}^k} \left( \frac{\partial \log p(s_i|e)}{\partial \phi_k(\bar{x}^k)} \right)_q \Delta\phi_k(\bar{x}^k) \\ &+ \frac{1}{2} \sum_{kl} \sum_{\bar{x}^k, \bar{y}^l} \left( \frac{\partial^2 \log p(s_i|e)}{\partial \phi_k(\bar{x}^k)\partial \phi_l(\bar{y}^l)} \right)_q \Delta\phi_k(\bar{x}^k)\Delta\phi_l(\bar{y}^l) \\ &+ \text{ higher order terms} \end{aligned} \tag{6}$$

The differentials are evaluated in the factorized distribution $q$. The left-hand size of Eq. 6 is zero because of Eq. 5 and we solve for $q(s_i)$. This factorized distribution gives the desired marginals up to the order of the expansion of $\Delta \log p(s_i|e)$.

It is straightforward to compute the derivatives:

$$
\begin{aligned}
\frac{\partial \log p(s_i|e)}{\partial \phi_k(\bar{x}^k)} &= p(\bar{x}^k|s_i, e) - p(\bar{x}^k|e) \\
\frac{\partial^2 \log p(s_i|e)}{\partial \phi_k(\bar{x}^k)\partial \phi_l(\bar{y}^l)} &= p(\bar{x}^k, \bar{y}^l|s_i, e) - p(\bar{x}^k, \bar{y}^l|e) \\
&\quad - p(\bar{x}^k|s_i, e)p(\bar{y}^l|s_i, e) + p(\bar{x}^k|e)p(\bar{y}^l|e)
\end{aligned} \tag{7}
$$

We introduce the notation $\langle\ldots\rangle_{s_i}$ and $\langle\ldots\rangle$ as the expectation values with respect to the factorized distributions $q(x|s_i, e)$ and $q(x|e)$, respectively. We define $\langle\langle\ldots\rangle\rangle_{s_i} \equiv \langle\ldots\rangle_{s_i} - \langle\ldots\rangle$. We obtain

$$
\begin{aligned}
\Delta \log p(s_i|e) &= \sum_k \langle\langle\Delta\phi_k\rangle\rangle_{s_i} \\
&+ \frac{1}{2}\sum_{k,l}\left(\langle\langle\Delta\phi_k\Delta\phi_l\rangle\rangle_{s_i} - \langle\Delta\phi_k\rangle_{s_i}\langle\Delta\phi_l\rangle_{s_i} + \langle\Delta\phi_k\rangle\langle\Delta\phi_l\rangle\right) \\
&+ \text{higher order terms}
\end{aligned} \tag{8}
$$

To first order, setting Eq. 8 equal to zero we obtain

$$
0 = \sum_k \langle\langle\Delta\phi_k\rangle\rangle_{s_i} = \langle\log p(x)\rangle_{s_i} - \log q(s_i) + \text{const.,} \tag{9}
$$

where we have absorbed all terms independent of $i$ into a constant. Thus, we find the solution

$$
q(s_i) = \frac{1}{Z_i}\exp\left(\langle\log p(x)\rangle_{s_i}\right) \tag{10}
$$

in which the constants $Z_i$ follow from normalisation. The first order term is equivalent to the standard mean field equations, obtained from Jensens' inequality.

The correction with second order terms is obtained in the same way, again dropping terms independent of $i$:

$$
q(s_i) = \frac{1}{Z_i}\exp\left(\langle\log p(x)\rangle_{s_i} + \frac{1}{2}\sum_{k,l}\left(\langle\Delta\phi_k\Delta\phi_l\rangle_{s_i} - \langle\Delta\phi_k\rangle_{s_i}\langle\Delta\phi_l\rangle_{s_i}\right)\right) \tag{11}
$$

were, again, the constants $Z_i$ follow from normalisation. These equations, which form the main result of this paper, are generalization of the mean field equations with TAP corrections for directed graphical models. Both left and right-hand size of Eqs. 10 and 11 depend on the unknown probability distribution $q(s)$ and can be solved by fixed point iteration.

## 3 Complexity and single-overlap graphs

The complexity of the first order equations Eq. 10 is exponential in the number of variables in the potentials $\phi_k$ of $P$: if the maximal clique size is $c$, then for each $i$ we need of the order of $n_i \exp(c)$ computations, where $n_i$ is the number of cliques that contain node $i$.

The second term scales worse, since one must compute averages over the union of two overlapping cliques and because of the double sum. However, things are not so bad. First

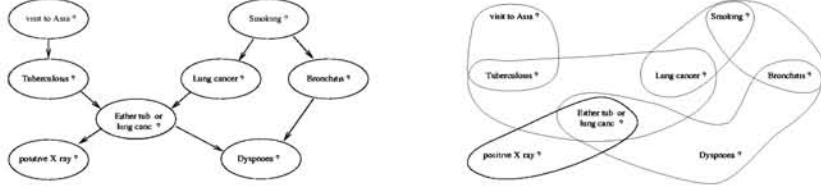

Figure 1: An example of a single-overlap graph. Left: The chest clinic model (ASIA)(Lauritzen and Spiegelhalter, 1988). Right: nodes within one potential a re grouped together, showing that potentials share at most one node.

of all, notice that the sum over $k$ and $l$ can be restricted to overlapping cliques ($k \cap l \neq \emptyset$) and that $i$ must be in either $k$ or $l$ or both ($i \in k \cup l$). Denote by $n^k$ the number of cliques that have at least one variable in common with clique $k$ and denote by $n_{\text{overlap}} = \max_k n_k$. Then, the sum over $k$ and $l$ contains not more than $n_i n_{\text{overlap}}$ terms.

Each term is an average over the union of two cliques, which can be worse case of size $2c-1$ (when only one variable is shared). However, since $\langle \Delta\phi_k \Delta\phi_l \rangle_{s_i} = \langle \langle \Delta\phi_k \rangle_{k \cap l} \Delta\phi_l \rangle_{s_i}$ ($\langle \cdot \rangle_{k \cap l}$ means expectation wrt $q$ conditioned on the variables in $k \cap l$) we can precompute $\langle \Delta\phi_k \rangle_{k \cap l}$ for all pairs of overlapping cliques $k, l$, for all states in $k \cap l$. Therefore, the worse case complexity of the second order term is less than $n_i n_{\text{overlap}} \exp(c)$. Thus, we see that the second order method has the same exponential complexity as the first order method, but with a different polynomial prefactor. Therefore, the first or second order method can be applied to directed graphical models as long as the number of parents is reasonably small.

The fact that the second order term has a worse complexity than the first order term is in contrast to Boltzmann machines, in which the TAP approximation has the same complexity as the standard mean field approximation. This phenomenon also occurs for a special class of DAGs, which we call single-overlap graphs. These are graphs in which the potentials $\phi_k$ share at most one node. Figure 1 shows an example of a single-overlap graph.

For single overlap graphs, we can use the first order result Eq. 9 to simplify the second order correction. The derivation rather tedious and we just present the result

$$q(s_i) = \frac{1}{Z_i} \exp\left( \langle \log p(x) \rangle_{s_i} + \frac{1}{2} \sum_{l, i \in l} \left( \langle (\Delta\phi_l)^2 \rangle_{s_i} - \langle \Delta\phi_l \rangle_{s_i}^2 \right) \right.$$

$$\left. - \sum_{l, i \in l} \sum_{j \neq i} \langle \langle \langle \Delta\phi_l \rangle \rangle_{s_j} \Delta\phi_l \rangle_{s_i} \right) \tag{12}$$

which has a complexity that is of order $n_i(c-1)\exp(c)$. For probability distributions with many small potentials that share nodes with many other potentials, Eq. 12 is more efficient than Eq. 11. For instance, for Boltzmann Machines $n_i = n_{\text{overlap}} = n - 1$ and $c = 2$. In this case, Eq. 12 is identical to the TAP equations (Thouless et al., 1977).

## 4   Sigmoid belief networks

In this section, we consider sigmoid belief networks as an interesting class of directed graphical models. The reason is, that one can expand in terms of the couplings instead of the potentials which is more efficient. The sigmoid belief network is defined as

$$p(x) = \prod_i \sigma(x_i h_i), \tag{13}$$

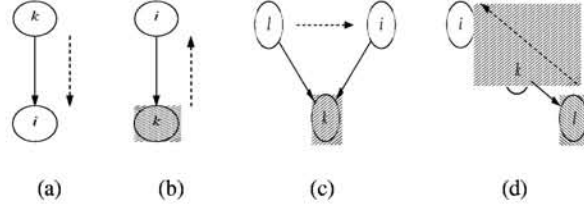

(a)          (b)          (c)          (d)

Figure 2: Interpretation of different interaction terms appearing in Eq. 16. The open and shaded nodes are hidden and evidence nodes, respectively (except in (a), where $k$ can be any node). Solid arrows indicate the graphical structure in the network. Dashed arrows indicate interaction terms that appear in Eq. 16.

where $\sigma(x) = (1 + \exp(-2x))^{-1}$, $x_i = \pm 1$ and $h_i$ is the local field: $h_i(x) = \sum_{j=1}^{n} w_{ij}x_j + \theta_i$.

We separate the variables in evidence variables $e$ and hidden variables $s$: $x = (s, e)$. When couplings from hidden nodes to either hidden or evidence nodes are zero, $w_{ij} = 0$, $i \in e, s$ and $j \in s$, the probability distributions $p(s|e)$ and $p(e)$ reduce to

$$p(s|e) \quad \rightarrow \quad q(s) = \prod_{i \in s} \sigma\left(s_i \theta_i^q\right) \tag{14}$$

$$p(e) \quad \rightarrow \quad r(e) = \prod_{i \in e} \sigma\left(e_i \theta_i^q\right) \tag{15}$$

where $\theta_i^q = \sum_{j \in e} w_{ij}e_j + \theta_i$ depends on the evidence.

We expand to second order around this tractable distribution and obtain

$$
\begin{aligned}
m_i \quad = \quad \tanh \Bigg( &\sum_{k \in s,e} m_j w_{ik} + \theta_i + 2 \sum_{k \in e} r(-e_k)e_k w_{ki} - m_i \sum_{k \in s}(1 - m_k^2)w_{ik}^2 \\
&+ 4m_i \sum_{k \in e} r(e_k)r(-e_k)w_{ki}^2 - 4 \sum_{k \in e, l \in s} r(e_k)r(-e_k)m_l w_{kl}w_{ki} \\
&+ 2 \sum_{k \in s, l \in e}(1 - m_k^2)r(-e_l)e_l w_{lk}w_{ki} \Bigg)
\end{aligned}
\tag{16}
$$

with $m_i = \langle s_i \rangle_q \approx \langle s_i \rangle_p$ and $r$ is given by Eq. 15.

The different terms that appear in this equation can be easily interpreted. The first term describes the lowest order forward influence on node $i$ from its parents. Parents can be either evidence or hidden nodes (fig. 2a). The second term is the bias $\theta_i$. The third term describes to lowest order the effect of Bayes' rule: it affects $m_i$ such that the observed evidence on its children becomes most probable (fig. 2b). Note, that this term is absent when the evidence is explained by the evidence nodes themselves: $r(e_k) = 1$. The fourth and fifth terms are the quadratic contributions to the first and third terms, respectively. The sixth term describes 'explaining away'. It describes the effect of hidden node $l$ on node $i$, when both have a common observed child $k$ (fig. 2c). The last term describes the effect on node $i$ when its grandchild is observed (fig. 2d).

Note, that these equations are different from Eq. 10. When one applies Eq. 10 to sigmoid belief networks, one requires additional approximations to compute $\langle \log \sigma(x_i h_i) \rangle$ (Saul et al., 1996).

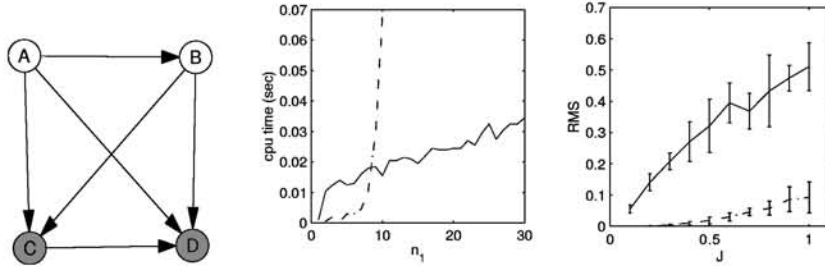

Figure 3: Second order approximation for fully connected sigmoid belief network of $n$ nodes. a) nodes $1, \dots, n_1$ are hidden (white) and nodes $n_1 + 1, \dots, n$ are clamped (grey), $n_1 = n/2$; b) CPU time for exact inference (dashed) and second order approximation (solid) versus $n_1$ ($J = 0.5$); c) RMS of hidden node exact marginals (solid) and RMS error of second order approximation (dashed) versus coupling strength $J$, ($n_1 = 10$).

Since only feed-forward connections are present, one can order the nodes such that $w_{ij} = 0$ for $i < j$. Then the first order mean field equations can be solved in one single sweep starting with node 1. The full second order equations can be solved by iteration, starting with the first order solution.

## 5  Numerical results

We illustrate the theory with two toy problems. The first one is inference in Lauritzen's chest clinic model (ASIA), defined on 8 binary variables $x = \{A, T, S, L, B, E, X, D\}$ (see figure 1, and (Lauritzen and Spiegelhalter, 1988) for more details about the model). We computed exact marginals and approximate marginals using the approximating methods up to first (Eq. 10) and second order (Eq. 11), respectively. The approximate marginals are determined by sequential iteration of (10) and (11), starting at $q(x_i) = 0.5$ for all variables $i$. The maximal error in the marginals using the first and second order method is 0.213 and 0.061, respectively. We verified that the single-overlap expression Eq. 12 gave similar results.

In fig. 3, we assess the accuracy and CPU time of the second order approximation Eq. 16 for sigmoid belief networks. We generate random fully connected sigmoid belief networks with $w_{ij}$ from a normal distribution with mean zero and variance $J^2/n$ and $\theta_i = 0$. We observe in fig. 3b that the computation time is very fast: For $n_1 = 500$, we have obtained convergence in 37 second on a Pentium 300 Mhz processor. The accuracy of the method depends on the size of the weights and is computed for a network of $n_1 = 10$ (fig. 3c). In (Kappen and Wiegerinck, 2001), we compare this approach to Saul's variational approach (Saul et al., 1996) and show that our approach is much faster and slightly more accurate.

# 6 Discussion

In this paper, we computed a second order mean field approximation for directed graphical models. We show that the second order approximation gives a significant improvement over the first order result. The method does not use explicitly that the graph is directed. Therefore, the result is equally valid for Markov graphs.

The complexity of the first and second order approximation is of $\mathcal{O}(n_i \exp(c))$ and $\mathcal{O}(n_i n_{\text{overlap}} \exp(c))$, respectively, with $c$ the number of variables in the largest potential. For single-overlap graphs, one can rewrite the second order equation such that the computational complexity reduces to $\mathcal{O}(n_i(c-1)\exp(c))$. Boltzmann machines and the Asia network are examples of single-overlap graphs.

For large $c$, additional approximations are required, as was proposed by (Saul et al., 1996) for the first order mean field equations. It is evident, that such additional approximations are then also required for the second order mean field equations.

It has been reported (Barber and Wiegerinck, 1999; Wiegerinck and Kappen, 1999) that similar numerical improvements can be obtained by using a very different approach, which is to use an approximating distribution $q$ that is not factorized, but still tractable. A promising way to proceed is therefore to combine both approaches and to do a second order expansion aroud a manifold of distributions with non-factorized yet tractable distributions. In this approach the sufficient statistics of the tractable structure is expanded, rather than the marginal probabilities.

### Acknowledgments

This research was supported in part by the Dutch Technology Foundation (STW).

# References

Barber, D. and Wiegerinck, W. (1999). Tractable variational structures for approximating graphical models. In Kearns, M., Solla, S., and Cohn, D., editors, *Advances in Neural Information Processing Systems*, volume 11 of *Advances in Neural Information Processing Systems*, pages 183–189. MIT Press.

Kappen, H. and Rodríguez, F. (1998). Efficient learning in Boltzmann Machines using linear response theory. *Neural Computation*, 10:1137–1156.

Kappen, H. and Spanjers, J. (1999). Mean field theory for asymmetric neural networks. *Physical Review E*, 61:5658–5663.

Kappen, H. and Wiegerinck, W. (2001). Mean field theory for graphical models. In Saad, D. and Opper, M., editors, *Advanced mean field theory*. MIT Press.

Lauritzen, S. and Spiegelhalter, D. (1988). Local computations with probabilties on graphical structures and their application to expert systems. *J. Royal Statistical society B*, 50:154–227.

Plefka, T. (1982). Convergence condition of the TAP equation for the infinite-range Ising spin glass model. *Journal of Physics A*, 15:1971–1978.

Saul, L., Jaakkola, T., and Jordan, M. (1996). Mean field theory for sigmoid belief networks. *Journal of artificial intelligence research*, 4:61–76.

Thouless, D., Anderson, P., and Palmer, R. (1977). Solution of 'Solvable Model of a Spin Glass'. *Phil. Mag.*, 35:593–601.

Wiegerinck, W. and Kappen, H. (1999). Approximations of bayesian networks through kl minimisation. *New Generation Computing*, 18:167–175.
